# Collective Oscillations in the Visual Cortex

**Daniel Kammen & Christof Koch**
Computation and Neural Systems
Caltech 216-76
Pasadena, CA 91125

**Philip J. Holmes**
Dept. of Theor. & Applied Mechanics
Cornell University
Ithaca, NY 14853

## ABSTRACT

The firing patterns of populations of cells in the cat visual cortex can exhibit oscillatory responses in the range of $35 - 85\ Hz$. Furthermore, groups of neurons many $mm$'s apart can be highly synchronized as long as the cells have similar orientation tuning. We investigate two basic network architectures that incorporate either nearest-neighbor or global feedback interactions and conclude that non-local feedback plays a fundamental role in the initial synchronization and dynamic stability of the oscillations.

## 1   INTRODUCTION

$40 - 60\ Hz$ oscillations have long been reported in the rat and rabbit olfactory bulb and cortex on the basis of single- and multi-unit recordings as well as EEG activity (Freeman, 1972; Wilson & Bower 1990). Recently, two groups (Eckhorn *et al.*, 1988 and Gray *et al.*, 1989) have reported highly synchronized, stimulus specific oscillations in the $35 - 85\ Hz$ range in areas 17, 18 and PMLS of anesthetized as well as awake cats. Neurons with similar orientation tuning up to $7\ mm$ apart show phase-locked oscillations, with a phase shift of less than $3\ msec$. We address here the computational architecture necessary to subserve this process by investigating to what extent two neuronal architectures, nearest-neighbor coupling and feedback from a central "comparator", can synchronize neuronal oscillations in a robust and rapid manner.

It was argued in earlier work on central pattern generators (Cohen *et al.*, 1982), that in studying coupling effects among large populations of oscillating neurons, one can ignore the details of individual oscillators and represent each one by a single periodic variable: its *phase*. Our approach assumes a population of neuronal oscillators, firing repetitively in response to synaptic input. Each cell (or group of tightly electrically coupled cells) has an associated variable representing the membrane potential. In particular, when $\theta_i = \pi$, an action potential is generated and the phase is reset to its initial value (in our case to $-\pi$). The number of times per unit time $\theta_i$ passes through $\pi$, i.e. $d\theta_i/dt$, is then proportional to the firing frequency of the neuron. For a network of $n+1$ such oscillators, our basic model is

$$\frac{d\theta_i}{dt} = \omega_i + f_i(\theta_0, \theta_1, ..., \theta_n), \qquad (1)$$

where $\omega_i$ represents the synaptic input to neuron $i$ and $f$, a function of the phases, represents the coupling within the network. Each oscillator $i$ in isolation (i.e. with $f_i = 0$), exhibits asymptotically stable periodic oscillations; that is, if the input is changed the oscillator will rapidly adjust to a new firing rate. In our model $\omega_i$ is assumed to derive from neurons in the lateral geniculate nucleus (LGN) and is purely excitatory.

## 2 FREQUENCY AND PHASE LOCKING

Any realistic model of the observed, highly synchronized, oscillations must account for the fact that the individual neurons oscillate at different frequencies in isolation. This is due to variations in the synaptic input, $\omega_i$, as well as in the intrinsic properties of the cells. We will contrast the abilities of two markedly different network architectures to synchronize these oscillations. The "chain" model (Fig. 1 top ) consists of a one-dimensional array of oscillators connected to their nearest neighbors, while in the alternative "comparator" model (Fig. 1 middle), an array of neurons project to a single unit, where the phases are averaged (i.e. $(1/n)\sum_{i=0}^{n}\theta_i(t)$). This average is then feed back to every neuron in the network. In the continuum limit (on the unit interval) with all $f_i = f$ being identical, the two models are

$$(Chain \quad Model) \qquad \frac{\partial\theta(x,t)}{\partial t} = \omega(x) + \frac{1}{n}\frac{\partial f}{\partial x}(\phi) \qquad (2)$$

$$(Comparator \quad Model) \qquad \frac{\partial\theta(x,t)}{\partial t} = \omega(x) + f(\theta(x,t) - \int_0^1 \theta(s,t)ds), \quad (3)$$

where $0 \leq x \leq 1$ and $\phi$ is the phase gradient, $\phi = \frac{1}{n}\frac{\partial\theta}{\partial x}$. In the chain model, we require that $f$ be an odd function (for simplicity of analysis only) while our analysis of the comparator model holds for any continuous function $f$. We use two spatially separated "spots" of width $\delta$ and amplitude $\alpha$ as visual input (Fig. 1 bottom). This pattern was chosen as a simple version of the double-bar stimulus that (Gray *et al.* 1989) found to evoke coherent oscillatory activity in widely separated populations of visual cortical cells.

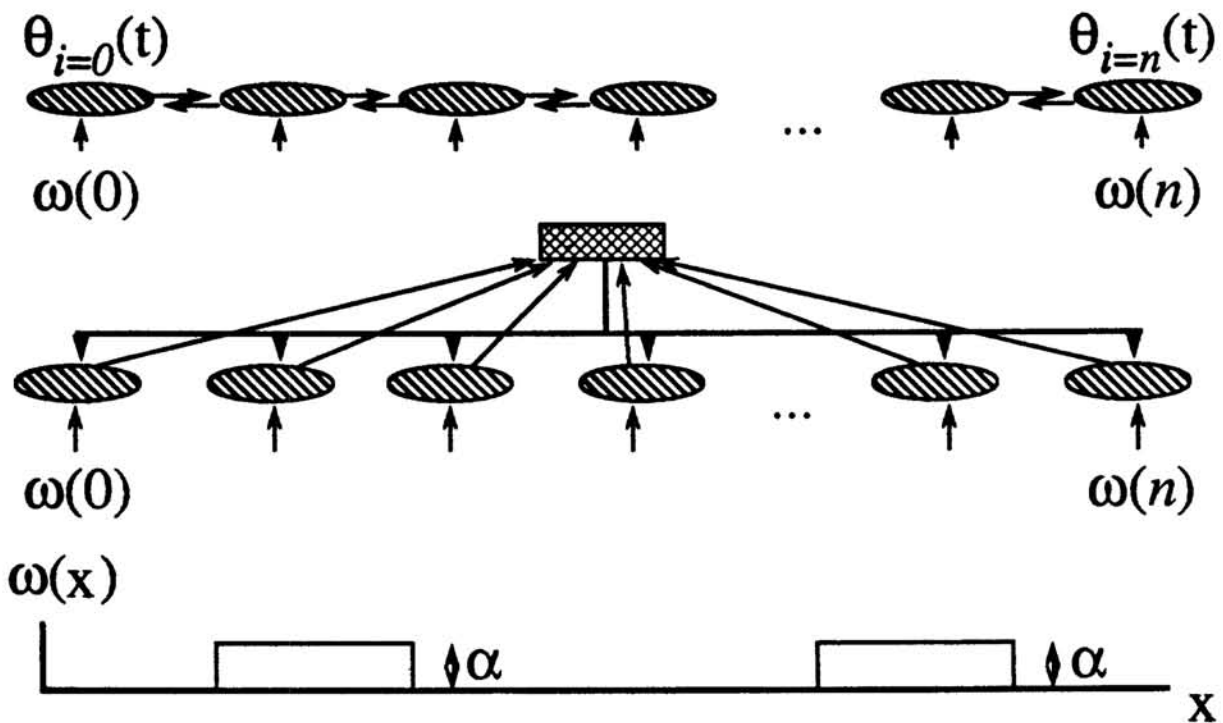

**Figure 1:** The linear chain (top) and comparator (middle) architectures. The spatial pattern of inputs is indicated by $\omega_i(x)$. See equs. 2 & 3 for a mathematical description of the models. The "two spot" input is shown at bottom and represents two parts of a perceptually extended figure.

We determine under what circumstances the chain model will develop frequency-locked solutions, such that every oscillator fires at the same frequency (but not necessarily at the same time), i.e. $\partial^2\theta/\partial x\partial t \equiv 0$. We prove (Kammen, *et al.* 1990) that frequency-locked solutions exist as long as $|n(\overline{\omega}x - \int_0^x \omega(s)ds)|$ does not exceed the maximal value of $f$, $f_{\max}$ (with $\overline{\omega} = \int_0^1 \omega(s)ds$ the mean excitation level). Thus, if the excitation is too irregular or the chain too long ($n \gg 1$), we will not find frequency-locked solutions. Phase coherence between the excited regions is *not* generally maintained and is, in fact, strongly a function of the initial conditions. Another feature of the chain model is that the onset of frequency locking is slow and takes time of order $\sqrt{n}$.

The location of the stimulus has no effect on phase relationships in the comparator model due to the global nature of the feedback. The comparator model exhibits two distinct regimes of behavior depending on the amplitude of the input, $\alpha$. In the case of the two spot input (Fig. 1 bottom), if $\alpha$ is small, all neurons will frequency-lock regardless of location, that is units responding to both the "figure" and the background ("ground") will oscillate at the same frequency. They will, however, fire at different times, with $\theta_{fig} \neq \theta_{gnd}$. If $\alpha$ is above a critical threshold, the units responding to the "figure" will decouple in frequency as well as phase from the background while still maintaining internal phase coherency. Phase gradients *never* exist within the excited groups, no matter what the input amplitude.

We numerically simulated the chain and comparator models with the two spot input for the coupling function $f(\eta) = sin(\eta)$. Additive Gaussian noise was included in the input, $\omega_i$. Our analytical results were confirmed; frequency and phase gradients were always present in the chain model (Fig. 2A) even though the coupling strength was ten times greater than that of the comparator model. In the comparator network small excitation levels led to frequency-locking along the entire array and to phase-coupled activity within the illuminated areas (Fig. 2B), while large excitation levels led to phase and frequency decoupling between the "figure" and the "background" (Fig. 2C). The excited regions in the comparator settle very rapidly – within 2 to 3 cycles – into phase-locked activity with small phase-delays. The chain model, on the other hand, exhibits strong sensitivity to initial conditions as well as a very slow approach to coherence that is still not complete even after 50 cycles (See Fig. 2).

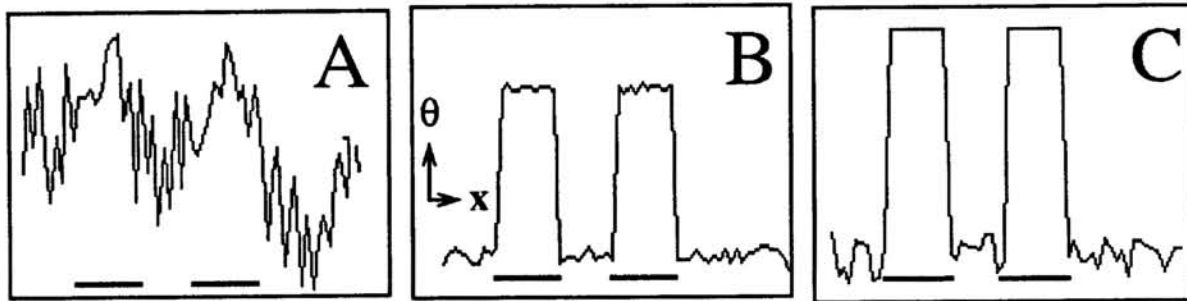

**Figure 2:** The phase portrait of the chain (A), weak (B) and strongly (C) excited comparator networks after 50 cycles. The input, indicated by the horizontal lines, is the two spot pattern. Note that the central, unstimulated, region in the chain model has been "dragged along" by the flanking excited regions.

## 3   STABILITY ANALYSIS

Perhaps the most intriguing aspect of the oscillations concerns the role that they may play in cortical information processing and the labeling of cells responding to a single perceptual object. To be useful in object coding, the oscillations must exhibit some degree of noise tolerance both in the input signal and in the stability of the population to variation in the firing times of individual cells.

The degree to which input noise to individual neurons disrupts the synchronization of the population is determined by the ratio $\frac{input\ noise}{coupling\ strength} = \frac{\omega_i(t)}{f(\cdot)}$. For small perturbations, $\omega(t) = \omega_0 + \epsilon(t)$, the action of the feedback, from the nearest neighbors in the chain and from the entire network in the comparator, will compensate for the noise and the neuron will maintain coherence with the excited population. As $\epsilon$ is increased first phase and then frequency coherence will be lost.

In Fig. 3 we compare the dynamical stability of the chain and comparator models. In each case the phase, $\theta$, of a unit receiving perturbated input is plotted as the deviation from the average phase, $\theta_0$, of all the excited units receiving input $\omega_0$. The chain in highly sensitive to noise: even 10% stochastic noise significantly perturbs the phase of the neuron. In the comparator model (Fig. 3B) noise must reach the

40% level to have a similar effect on the phase. As the noise increases above $0.30\omega_0$ even *frequency* coherence is lost in the chain model (broken error bars). Frequency coherence is maintained in the comparator for $\epsilon = 0.60\omega_0$.

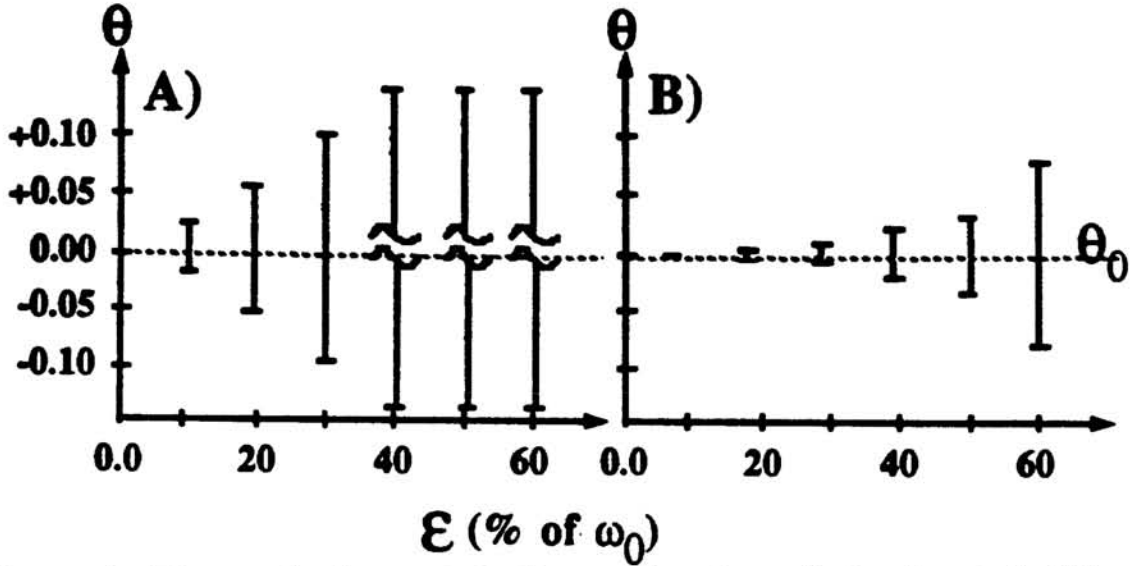

**Figure 3:** The result of a perturbation on the phase, $\theta$, for the chain (A) and comparator (B) models. The terminus of the error bars gives the resulting deviation from the unperturbed value. Broken bars indicate both phase and frequency decoupling.

The stability of the solutions of the comparator model to variability in the activity of individual neurons can easily be demonstrated. For simplicity consider the case of a single input of amplitude $\omega_1$ superposed on a background of amplitude $\omega_0$. The solutions in each region are:

$$\frac{d\theta_0}{dt} = \omega_0 + f\left(\frac{\theta_0 - \theta_1}{2}\right) \tag{4}$$

$$\frac{d\theta_1}{dt} = \omega_1 + f\left(\frac{\theta_1 - \theta_0}{2}\right). \tag{5}$$

We define the difference in the solutions to be $\phi(t) = \theta_1(t) - \theta_0(t)$ and $\Delta\omega = \omega_1 - \omega_0$. We then have an equation for the rate the solutions converge or diverge:

$$\frac{d\phi}{dt} = \Delta\omega + f(\frac{\phi}{2}) - f(-\frac{\phi}{2}). \tag{6}$$

If the solutions are stable (of constant velocity) then $d\theta_1/dt = d\theta_0/dt$ and $\theta_1 = \theta_0 + c$ with $c$ a constant. We then have the stable solution $\phi^* = c \; d\phi^*/dt = \Delta\omega + f(\frac{c}{2}) - f(-\frac{c}{2}) = 0$. Stability of the solutions can be seen by perturbing $\theta_1$ to $\theta_1 = \theta_0 + c + \epsilon$ with $|\epsilon| < 1$. The perturbed solution, $\phi = \phi^* + \epsilon$, has the derivative $d\phi/dt = d\epsilon/dt$. Developing $f(\phi)$ into a Taylor series around $\phi^*$ and neglecting terms on the order of $\epsilon^2$ and higher, we arrive at

$$\frac{d\epsilon}{dt} = \frac{\epsilon}{2}\left[f'(\frac{c}{2}) + f'(\frac{-c}{2})\right]. \tag{7}$$

If $f(\phi)$ is odd then $f'(\phi)$ is even, and eq. (7) reduces to

$$\frac{d\epsilon}{dt} = \epsilon f'(\frac{c}{2}). \tag{8}$$

Thus, if $f'(c/2) < 0$ the perturbations will decay to zero and the system will maintain phase locking within the excited regions.

## 4  THE FREQUENCY MODEL

The model discussed so far assumes that the feedback is only a function of the phases. In particular, this implies that the comparator computes the average phase across the population. Consider, however, a model where the feedback is proportional to the average firing frequency of a group of neurons. Let us therefore replace phase in the feedback function with firing frequency,

$$\frac{\partial\theta(x,t)}{\partial t} = \omega(x) + f\left(\frac{\partial\theta(x,t)}{\partial t} - \overline{\frac{\partial\theta(x,t)}{\partial t}}\right) \tag{9}$$

with $\overline{\frac{\partial\theta}{\partial t}} = \frac{\partial}{\partial t}\int_0^1 \theta(s,t)ds = \frac{\partial\bar\theta}{\partial t}$. This is a very special differential equation as can be seen by setting $v(x,t) = \partial\theta(x,t)/\partial t$. This yields an algebraic equation for $v$ with no explicit time dependency:

$$v(x) = w(x) + f(v(x) - \bar v(x)) \tag{10}$$

and, after an integration, we have,

$$\theta(x,t) = \int_0^t v(x)dt = v(x)t + \theta_0(x). \tag{11}$$

Thus, the phase relationships depend on the initial conditions, $\theta_0(x)$, and no phase locking occurs. While frequency locking only occurs for $\omega(x) = 0$ the feedback can lead to tight frequency coupling among the excited neurons.

Reformulating the chain model in terms of firing-frequencies, we have

$$\frac{\partial\theta(x,t)}{\partial t} = \frac{1}{n}\left(\frac{\partial\omega(x)}{\partial x} + \frac{1}{n}\frac{\partial^2}{\partial x^2}f\left(\frac{\partial\theta(x,t)}{\partial t}\right)\right) \tag{12}$$

under the assumption that $f(-x) = -f(x)$. With $\gamma(x,t) = \frac{\partial\phi(x,t)}{\partial t}$, we again arrive at a stationary algebraic equation

$$\gamma(x) = \frac{1}{n}\left(\frac{\partial w}{\partial x} + \frac{1}{n}\frac{\partial^2}{\partial x^2}f(\gamma(x))\right), \tag{13}$$

and

$$\phi(x,t) = \int_0^t \gamma(x)dt = \gamma(x)t + \phi_0(x) \tag{14}$$

In other words, the system will develop a time-dependent phase gradient. Frequency locked solutions of the sort $\frac{\partial\theta}{\partial t} = 0$ everywhere only occur if $\omega(x) = 0$ everywhere. Thus, the chain architecture leads to very static behavior, with little ability to either phase- or frequency-lock.

## 5  DISCUSSION

We have investigated the ability of two networks of relaxation oscillators with different connectivity patterns to synchronize their oscillations. Our investigation has been prompted by recent experimental results pertaining to the existence of frequency- and phase-locked oscillations in the mammalian visual cortex (Gray *et al.*, 1989; Eckhorn *et al.*, 1988). While these $35 - 85\,Hz$ oscillations are induced by the visual stimulus, usually a flashing or moving bar, they are not locked to the frequency of the stimulus. Most surprising is the finding that cells tuned to the same orientation, but separated by up to 7 $mm$, not only exhibit coherent oscillatory activity, but do so with a phase-shift of less than 3 $msec$ (Gray *et al.*, 1989).[1]

We have assumed the existence of a population of cortical oscillators, such as those reported in cortical slice preparations (Llinás, 1988; Chagnac-Amitai and Connors, 1989). The issue is then how such a population of oscillators can rapidly begin to fire in near total synchrony. Two neuronal architectures suggest themselves.

As a mechanism for establishing coherent oscillatory activity the comparator model is far superior to a nearest-neighbor model. The comparator rapidly (within $1 - 3$ cycles) achieves phase coherence, while the chain model exhibits a far slower onset of synchronization and is highly sensitive to the initial conditions. Once initiated, the oscillations in the two models exhibit markedly different stability characteristics. The diffusive nature of communication in the chain results in little ability to regulate the firing of individual units and consequently only highly homogeneous inputs will result in collective oscillations. The long-range connections present in the comparator, however, result in stable collective oscillations even in the presence of significant noise levels. Noise uniformly distributed about the mean firing level will have little effect due to the averaging performed by the comparator unit.

A more realistic model of the interconnection architecture of the cortex will certainly have to take both local as well as global neuronal pathways into account and the ever-present delays in cellular and network signal propagation (Kammen, *et al.*, 1990). Long range (up to 6 $mm$) lateral excitatory connections have been reported (Gilbert and Wiesel, 1983). However, their low conduction velocities ($\approx 1\ mm/msec$) would lead to significant phase-shifts in contrast to the data. While the cortical circuitry contains both local as well as global connection, our results imply that a cortical architecture with one or more "comparator" neurons driven by the averaged activity of the hypercolumnar cell populations is an attractive mechanism for synchronizing the observed oscillations.

We have also developed a model where the firing frequency, and not the phase is involved in the dynamics. Coding based on phase information requires that the cells track the time interval between incident spikes whereas the firing frequency is available as the raw spike rate. This computation can be readily implemented

neurobiologically and is entirely consistent with the known biophysics of cortical cells.

Von der Malsburg (1985) has argued that the temporal synchronization of groups of neurons labels perceptually distinct objects, subserving figure-ground segregation. Both firing frequency and inter-cell phase (timing) relationships of ensembles of neurons are potential channels to encode the signatures of various objects in the visual field. Perceptually distinct objects could be coded by groups of synchronized neurons, all locked to the same frequency with the groups only distinguished by their phase relationships. We do not believe, however, that phase is a robust enough variable to code this information across the cortex, A more robust scheme is one in which groups of synchronized neurons are locked at different firing frequencies.

## Acknowledgement

D.K. is a recipient of a Weizman Postdoctoral Fellowship. P.H. acknowledges support from the Sherman Fairchild Foundation and C.K. from the Air Force Office of Scientific Research, a NSF Presidential Young Investigator Award and from the James S. McDonnell Foundation. We would like to thank Francis Crick for useful comments and discussions.

## Footnotes

[1] Note that this result is obtained by averaging over many trials. The phase-shift for individual trial may possibly be larger, but could be randomly distributed from trial to trial around the origin.

## References

Chagnac-Amitai, Y. & Connors, B. W. (1989) *J. Neurophys.*, **62**, 1149.

Cohen, A. H., Holmes, P. J. & Rand R. H. (1982) *J. Math. Biol.* **3**, 345.

Eckhorn, R., Bauer, R., Jordan, W., Brosch, M., Kruse, W., Munk, M. & Reitboeck, H. J. (1988) *Biol. Cybern.*, **60**, 121.

Freeman, W.J. (1972) *J. Neurophysiol.* **35**, 762.

Gilbert, C. D. & T.N. Wiesel (1983) *J. Neurosci.* **3**, 1116.

Gray, C. M., König, P., Engel, A. K. & Singer, W. (1989) *Nature* **338**, 334.

Kammen, D. M., Koch, C. and Holmes, P. J. (1990) *Proc. Natl. Acad. Sci. USA*, submitted.

Kopell N. & Ermentrout, G. B. (1986) *Comm. Pure Appl. Math.* **39**, 623.

Llinás, R. R. (1988) *Science* **242**, 1654.

von der Malsburg, C. (1985) *Ber. Bunsenges Phys. Chem.*, **89**, 703.

Wilson, M. A. & Bower, J. (1990) *J. Neurophysiol.*, in press.